# Learning From Demonstration

**Stefan Schaal**
sschaal@cc.gatech.edu; http://www.cc.gatech.edu/fac/Stefan.Schaal

College of Computing, Georgia Tech, 801 Atlantic Drive, Atlanta, GA 30332-0280
ATR Human Information Processing, 2-2 Hikaridai, Seiko-cho, Soraku-gun, 619-02 Kyoto

## Abstract

By now it is widely accepted that learning a task from scratch, i.e., without any prior knowledge, is a daunting undertaking. Humans, however, rarely attempt to learn from scratch. They extract initial biases as well as strategies how to approach a learning problem from instructions and/or demonstrations of other humans. For learning control, this paper investigates how learning from demonstration can be applied in the context of reinforcement learning. We consider priming the $Q$-function, the value function, the policy, and the model of the task dynamics as possible areas where demonstrations can speed up learning. In general nonlinear learning problems, only model-based reinforcement learning shows significant speed-up after a demonstration, while in the special case of linear quadratic regulator (LQR) problems, all methods profit from the demonstration. In an implementation of pole balancing on a complex anthropomorphic robot arm, we demonstrate that, when facing the complexities of real signal processing, model-based reinforcement learning offers the most robustness for LQR problems. Using the suggested methods, the robot learns pole balancing in just a *single* trial after a 30 second long demonstration of the human instructor.

## 1. INTRODUCTION

Inductive supervised learning methods have reached a high level of sophistication. Given a data set and some prior information about its nature, a host of algorithms exist that can extract structure from this data by minimizing an error criterion. In learning control, however, the learning task is often less well defined. Here, the goal is to learn a policy, i.e., the appropriate actions in response to a perceived state, in order to steer a dynamical system to accomplish a task. As the task is usually described in terms of optimizing an arbitrary performance index, no direct training data exist which could be used to learn a controller in a supervised way. Even worse, the performance index may be defined over the long term behavior of the task, and a problem of temporal credit assignment arises in how to credit or blame actions in the past for the current performance. In such a setting, typical for reinforcement learning, learning a task from scratch can require a prohibitively time-consuming amount of exploration of the state-action space in order to find a good policy.

On the other hand, learning without prior knowledge seems to be an approach that is rarely taken in human and animal learning. Knowledge how to approach a new task can be transferred from previously learned tasks, and/or it can be extracted from the performance of a teacher. This opens the questions of how learning control can profit from these kinds of information in order to accomplish a new task more quickly. In this paper we will focus on learning from demonstration.

Learning from demonstration, also known as "programming by demonstration", "imitation learning", and "teaching by showing" received significant attention in automatic robot assembly over the last 20 years. The goal was to replace the time-consuming manual pro-

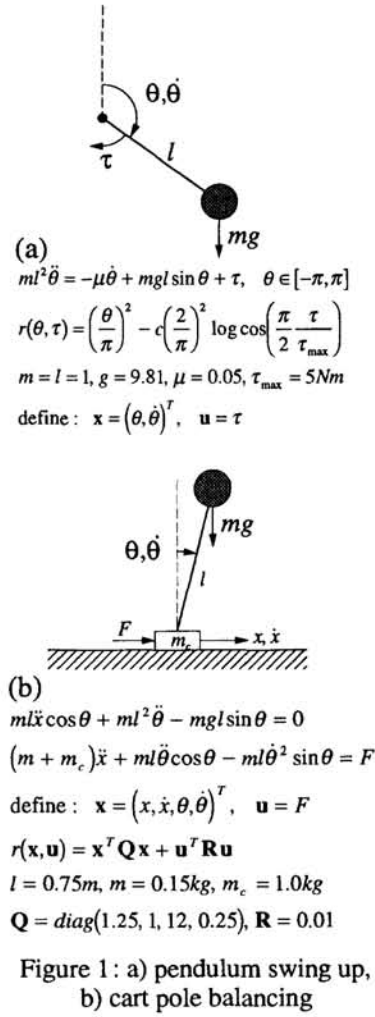

(a)

$$ml^2\ddot{\theta} = -\mu\dot{\theta} + mgl\sin\theta + \tau, \quad \theta \in [-\pi,\pi]$$

$$r(\theta,\tau) = \left(\frac{\theta}{\pi}\right)^2 - c\left(\frac{2}{\pi}\right)^2 \log\cos\left(\frac{\pi}{2}\frac{\tau}{\tau_{max}}\right)$$

$$m = l = 1, g = 9.81, \mu = 0.05, \tau_{max} = 5Nm$$

define: $\mathbf{x} = (\theta,\dot{\theta})^T, \quad \mathbf{u} = \tau$

(b)

$$ml\ddot{x}\cos\theta + ml^2\ddot{\theta} - mgl\sin\theta = 0$$

$$(m + m_c)\ddot{x} + ml\ddot{\theta}\cos\theta - ml\dot{\theta}^2\sin\theta = F$$

define: $\mathbf{x} = (x,\dot{x},\theta,\dot{\theta})^T, \quad \mathbf{u} = F$

$$r(\mathbf{x},\mathbf{u}) = \mathbf{x}^T\mathbf{Q}\mathbf{x} + \mathbf{u}^T\mathbf{R}\mathbf{u}$$

$$l = 0.75m, m = 0.15kg, m_c = 1.0kg$$

$$\mathbf{Q} = diag(1.25, 1, 12, 0.25), \mathbf{R} = 0.01$$

Figure 1: a) pendulum swing up, b) cart pole balancing

gramming of a robot by an automatic programming process, solely driven by showing the robot the assembly task by an expert. In concert with the main stream of Artificial Intelligence at the time, research was driven by symbolic approaches: the expert's demonstration was segmented into primitive assembly actions and spatial relationships between manipulator and environment, and subsequently submitted to symbolic reasoning processes (e.g., Lozano-Perez, 1982; Dufay & Latombe, 1983; Segre & DeJong, 1985). More recent approaches to programming by demonstration started to include more inductive learning components (e.g., Ikeuchi, 1993; Dillmann, Kaiser, & Ude, 1995). In the context of human skill learning, teaching by showing was investigated by Kawato, Gandolfo, Gomi, & Wada (1994) and Miyamoto et al. (1996) for a complex manipulation task to be learned by an anthropomorphic robot arm. An overview of several other projects can be found in Bakker & Kuniyoshi (1996).

In this paper, the focus lies on reinforcement learning and how learning from demonstration can be beneficial in this context. We divide reinforcement learning into two categories: reinforcement learning for nonlinear tasks (Section 2) and for (approximately) linear tasks (Section 3), and investigate how methods like *Q*-learning, value-function learning, and model-based reinforcement learning can profit from data from a demonstration. In Section 2.3, one example task, pole balancing, is placed in the context of using an actual, anthropomorphic robot to learn it, and we reconsider the applicability of learning from demonstration in this more complex situation.

## 2. REINFORCEMENT LEARNING FROM DEMONSTRATION

Two example tasks will be the basis of our investigation of learning from demonstration. The nonlinear task is the "pendulum swing-up with limited torque" (Atkeson, 1994; Doya, 1996), as shown in Figure 1a. The goal is to balance the pendulum in an upright position starting from hanging downward. As the maximal torque available is restricted such that the pendulum cannot be supported against gravity in all states, a "pumping" trajectory is necessary, similar as in the mountain car example of Moore (1991), but more delicately in its timing since building up too much momentum during pumping will overshoot the upright position. The (approximately) linear example, Figure 1b, is the well-known cart-pole balancing problem (Widrow & Smith, 1964; Barto, Sutton, & Anderson, 1983). For both tasks, the learner is given information about the one-step reward $r$ (Figure 1), and both tasks are formulated as continuous state and continuous action problems. The goal of each task is to find a policy which minimizes the infinite horizon discounted reward:

$$V(\mathbf{x}(t)) = \int_t^\infty e^{-\frac{(s-t)}{\tau}} r(\mathbf{x}(s),\mathbf{u}(s))ds \quad \text{or} \quad V(\mathbf{x}(t)) = \sum_{i=t}^\infty \gamma^{i-t} r(\mathbf{x}(i),\mathbf{u}(i)) \quad (1)$$

where the left hand equation is the continuous time formulation, while the right hand equation is the corresponding discrete time version, and where $\mathbf{x}$ and $\mathbf{u}$ denote a $n$-dimensional state vector and a $m$-dimensional command vector, respectively. For the Swing-Up, we assume that a teacher provided us with 5 successful trials starting from dif-

ferent initial conditions. Each trial consists of a time series of data vectors $(\theta, \dot{\theta}, \tau)$ sampled at 60Hz. For the Cart-Pole, we have a 30 second demonstration of successful balancing, represented as a 60Hz time series of data vectors $(x, \dot{x}, \theta, \dot{\theta}, F)$. How can these demonstrations be used to speed up reinforcement learning?

## 2.1  THE NONLINEAR TASK: SWING-UP

We applied reinforcement learning based on learning a value function (V-function) (Dyer & McReynolds, 1970) for the Swing-Up task, as the alternative method, Q–learning (Watkins, 1989), has yet received very limited research for continuous state-action spaces. The V–function assigns a scalar reward value $V(\mathbf{x}(t))$ to each state $\mathbf{x}$ such that the entire V–function fulfills the consistency equation:

$$V(\mathbf{x}(t)) = \arg\min_{\mathbf{u}(t)}\left(r(\mathbf{x}(t), \mathbf{u}(t)) + \gamma\, V(\mathbf{x}(t+1))\right) \qquad (2)$$

For clarity, this equation is given for a discrete state-action system; the continuous formulation can be found, e.g., in Doya (1996). The optimal policy, $\mathbf{u} = \pi(\mathbf{x})$, chooses the action $\mathbf{u}$ in state $\mathbf{x}$ such that (2) is fulfilled. Note that this computation involves an optimization step that includes knowledge of the subsequent state $\mathbf{x}(t+1)$. Hence, it requires a model of the dynamics of the controlled system, $\mathbf{x}(t+1)=f(\mathbf{x}(t),\mathbf{u}(t))$. From the viewpoint of learning from demonstration, V-function learning offers three candidates which can be primed from a demonstration: the value function $V(\mathbf{x})$, the policy $\pi(\mathbf{x})$, and the model $f(\mathbf{x},\mathbf{u})$.

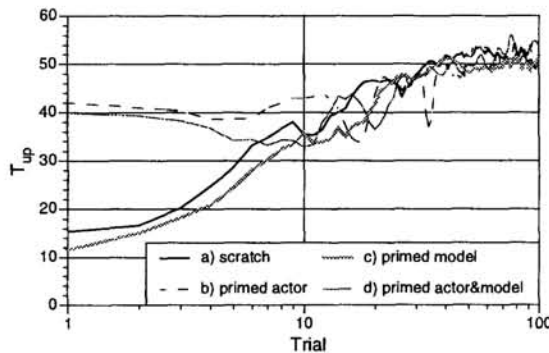

### 2.1.1  V-Learning

In order to assess the benefits of a demonstration for the Swing-Up, we implemented V–learning as suggested in Doya's (1996) continuous TD (CTD) learning algorithm. The V–function and the dynamics model were incrementally learned by a nonlinear function approximator, Receptive Field Weighted Regression (RFWR) (Schaal & Atkeson (1996)). Differing from Doya's (1996) implementation, we used the optimal action suggested by CTD to learn a model of the policy $\pi$ (an "actor" as in Barto et al. (1983)), again represented by RFWR. The following learning conditions were tested empirically:

Figure 2: Smoothed learning curves of the average of 10 learning trials for the learning conditions a) to d) (see text). Good performance is characterized by $T_{up} > 45s$; below this value the system is usually able to swing up properly but it does not know how to stop in the upright position.

a)  *Scratch*: Trial by trial learning of value function $V$, model $f$, and actor $\pi$ from scratch.
b)  *Primed Actor*: Initial training of $\pi$ from the demonstration, then trial by trial learning.
c)  *Primed Model*: Initial training of $f$ from the demonstration, then trial by trial learning.
d)  *Primed Actor&Model*: Priming of $\pi$ and $f$ as in b) and c), then trial by trial learning.

Figure 2 shows the results of learning the Swing-Up. Each trial lasted 60 seconds. The time $T_{up}$ the pole spent in the interval $\theta \in [-\pi/2, \pi/2]$ during each trial was taken as the performance measure (Doya, 1996). Comparing conditions a) and c), the results demonstrate that learning the pole model from the demonstration did not speed up learning. This is not surprising since learning the V–function is significantly more complicated than learning the model, such that the learning process is dominated by V–function learning. Interestingly, priming the actor from the demonstration had a significant effect on the initial performance (condition a) vs. b)). The system knew right away how to pump up the pendulum, but, in order to learn how to balance the pendulum in the upright position, it finally took the same amount of time as learning from scratch. This behavior is due to the

fact that, theoretically, the *V*–function can only be approximated correctly if the entire state-action space is explored densely. Only if the demonstration covered a large fraction of the entire state space one would expect that *V*–learning can profit from it. We also investigated using the demonstration to prime the *V*–function by itself or in combination with the other functions. The results were qualitatively the same as in shown in Figure 2: if the policy was included in the priming, the learning traces were like b) and d), otherwise like a) and c). Again, this is not totally surprising. Approximating a *V*-function is not just supervised learning as for π and *f*, it requires an iterative procedure to ensure the validity of (2) and amounts to a complicated nonstationary function approximation process. Given the limited amount of data from the demonstration, it is generally very unlikely to approximate a good value function.

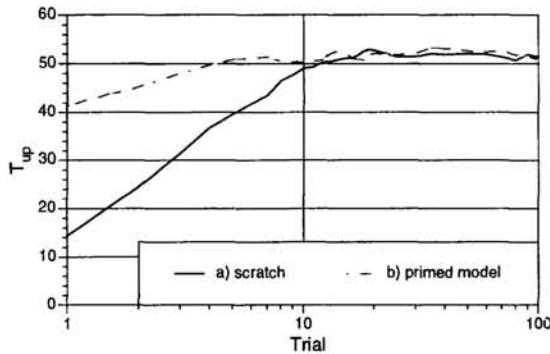

Figure 3: Smoothed learning curves of the average of 10 learning trials for the learning conditions a) and b) (see text) of the Swing-Up problem using "mental simulations". See Figure 2 for explanations how to interpret the graph.

### 2.1.2 Model-Based *V*-Learning

If learning a model *f* is required, one can make more powerful use of it. According to the certainty equivalence principle, *f* can substitute the real world, and planning can be run in "mental simulations" instead of interaction with the real world. In reinforcement learning, this idea was originally pursued by Sutton's (1990) DYNA algorithms for discrete state-action spaces. Here we will explore in how far a continuous version of DYNA, DYNA-CTD, can help in learning from demonstration. The only difference compared to CTD in Section 2.1.1 is that after every real trial, DYNA-CTD performs five "mental trials" in which the model of the dynamics acquired so far replaces the actual pole dynamics. Two learning conditions we be explored:

a)   *Scratch*: Trial by trial learning of *V*, model *f*, and policy π from scratch.
b)   *Primed Model*: Initial training of *f* from the demonstration, then trial by trial learning.

Figure 3 demonstrates that in contrast to *V*–learning in the previous section, learning from demonstration can make a significant difference now: after the demonstration, it only takes about 2-3 trials to accomplish a good swing-up with stable balancing, indicated by $T_{up} > 45s$. Note that also learning from scratch is significantly faster than in Figure 2.

## 2.2 THE LINEAR TASK: CART-POLE BALANCING

One might argue that applying reinforcement learning from demonstration to the Swing-Up task is premature, since reinforcement learning with nonlinear function approximators has yet to obtain appropriate scientific understanding. Thus, in this section we turn to an easier task: the cart-pole balancer. The task is approximately linear if the pole is started in a close to upright position, and the problem has been well studied in the dynamic programming literature in the context of linear quadratic regulation (LQR) (Dyer & McReynolds, 1970).

### 2.2.1 Q–Learning

In contrast to *V*-learning, *Q*–learning (Watkins, 1989; Singh & Sutton, 1996) learns a more complicated value function, $Q(\mathbf{x}, \mathbf{u})$, which depends both on the state and the command. The analogue of the consistency equation (2) for *Q*–learning is:

$$Q(\mathbf{x}(t), \mathbf{u}(t)) = r(\mathbf{x}(t), \mathbf{u}(t)) + \gamma \underset{\mathbf{u}(t+1)}{\arg \min}\left(Q(\mathbf{x}(t+1), \mathbf{u}(t+1))\right) \qquad (3)$$

At every state $\mathbf{x}$, picking the action $\mathbf{u}$ which minimizes $Q$ is the optimal action under the reward function (1). As an advantage, evaluating the $Q$–function to find the optimal policy *does not* require a model the dynamical system $f$ that is to be controlled; only the value of the one-step reward $r$ is needed. For learning from demonstration, priming the Q-function and/or the policy are the two candidates to speed up learning.

For LQR problems, Bradtke (1993) suggested a $Q$–learning method that is ideally suited for learning from demonstration, based on extracting a policy. He observed that for LQR the $Q$–function is quadratic in the states and commands:

$$Q(\mathbf{x},\mathbf{u}) = \left[\mathbf{x}^T, \mathbf{u}^T\right]\begin{bmatrix}\mathbf{H}_{11} & \mathbf{H}_{12}\\ \mathbf{H}_{21} & \mathbf{H}_{22}\end{bmatrix}\left[\mathbf{x}^T, \mathbf{u}^T\right]^T, \ \mathbf{H}_{11} = n \times n, \ \mathbf{H}_{22} = m \times m, \ \mathbf{H}_{12} = \mathbf{H}_{21}^T = n \times m \quad (4)$$

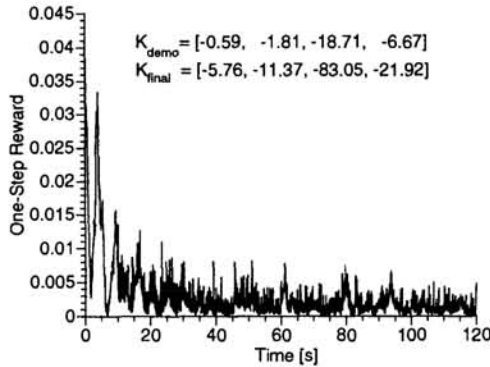

Figure 4: Typical learning curve of a noisy simulation of the cart-pole balancer using $Q$-learning. The graph shows the value of the one-step reward over time for the first learning trial. The pole is never dropped.

and that the (linear) policy, represented as a gain matrix $\mathbf{K}$, can be extracted from (4) as:

$$u_{opt} = -\mathbf{K}\mathbf{x} = -\mathbf{H}_{22}^{-1}\mathbf{H}_{21}\mathbf{x} \quad (5)$$

Conversely, given a stabilizing initial policy $\mathbf{K}_{demo}$, the current $Q$–function can be approximated by a recursive least squares procedure, and it can be optimized by a policy iteration process with guaranteed convergence (Bradkte, 1993). As a demonstration allows one to extract an initial policy $\mathbf{K}_{demo}$ by linearly regressing the observed command $\mathbf{u}$ against the corresponding observed states $\mathbf{x}$, one-shot learning of pole balancing is achievable. As shown in Figure 4, after about 120 seconds (12 policy iteration steps), the policy is basically indistinguishable from the optimal policy. A caveat of this $Q$–learning, however, is that it cannot not learn *without* a *stabilizing* initial policy.

### 2.2.2 Model-based $V$–Learning

Learning an LQR task by learning the $V$-function is one of the classic forms of dynamic programming (Dyer & McReynolds, 1970). Using a stabilizing initial policy $\mathbf{K}_{demo}$, the current $V$–function can be approximated by recursive least squares in analogy with Bradtke (1993). Similarly as for $\mathbf{K}_{demo}$, a (linear) model $f_{demo}$ of the cart-pole dynamics can be extracted from a demonstration by linear regression of the cart-pole state $\mathbf{x}(t)$ vs. the previous state and command vector $(\mathbf{x}(t-1), \mathbf{u}(t-1))$, and the model can be refined with every new data point experienced during learning. The policy update becomes:

$$\mathbf{K} = \gamma\left(R + \gamma \mathbf{B}^T\mathbf{H}\mathbf{B}\right)^{-1}\mathbf{B}^T\mathbf{H}\mathbf{A}, \text{ where } V(\mathbf{x}) = \mathbf{x}^T\mathbf{H}\mathbf{x}, f_{demo} = [\mathbf{A}\,\mathbf{B}], \mathbf{A} = n \times n, \mathbf{B} = n \times m \quad (6)$$

Thus, a similar process as in Bradtke (1993) can be used to find the optimal policy $\mathbf{K}$, and the system accomplishes one shot learning, qualitatively indistinguishable from Figure 4.

Again, as pointed out in Section 2.1.2, one can make more efficient use of the learned model by performing mental simulations. Given the model $f_{demo}$, the policy $\mathbf{K}$ can be calculated by off-line policy iteration from an initial estimate of $\mathbf{H}$, e.g., taken to be the identity matrix (Dyer & McReynolds, 1970). Thus, no initial (stabilizing) policy is required, but rather an estimate of the task dynamics. Also this method achieves one shot learning.

### 2.3 POLE BALANCING WITH AN ACTUAL ROBOT

As a result of the previous section, it seems that there are no real performance differences between $V$-learning, $Q$-learning, and model-based $V$-learning for LQR problems. To explore the usefulness of these methods in a more realistic framework, we implemented

learning from demonstration of pole balancing on an anthropomorphic robot arm. The robot is equipped with a 60 Hz video-based stereo vision. The pole is marked by two color blobs which can be tracked in real-time. A 30 second long demonstration of pole balancing was is provided by a human standing in front of the two robot cameras.

There are a few crucial differences in comparison with the simulations. First, as the demonstration is vision-based, only kinematic variables can be extracted from the demonstration. Second, visual signal processing has about 120ms time delay. Third, a command given to the robot is not executed with very high accuracy due to unknown nonlinearities of the robot. And lastly, humans use internal state for pole balancing, i.e., their policy is partially based on non-observable variables. These issues have the following impact:

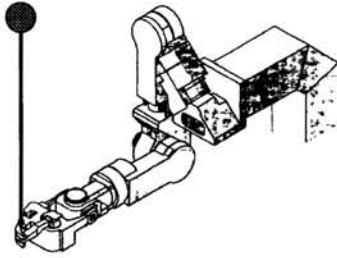

*Kinematic Variables*: In this implementation, the robot arm replaces the cart of the Cart-Pole problem. Since we have an estimate of the inverse dynamics and inverse kinematics of the arm, we can use the acceleration of the finger in Cartesian space as command input to the task. The arm is also much heavier than the pole which allows us to neglect the interaction forces the pole exerts on the arm. Thus, the pole balancing dynamics of Figure 1b can be reformulated as:

$$uml\cos\theta + \ddot{\theta}ml^2 - mgl\sin\theta = 0, \quad \ddot{x} = u \quad\quad (7)$$

Figure 5: Sketch of SARCOS anthropomorphic robot arm

All variables in this equation can be extracted from a demonstration. We omit the 3D extension of these equations.

*Delayed Visual Information*: There are two possibilities of dealing with delayed variables. Either the state of the system is augmented by delayed commands corresponding to $7*1/60s \approx 120s$ delay time, $\mathbf{x}^T = (x, \dot{x}, \theta, \dot{\theta}, u_{t-1}, u_{t-2}, \ldots, u_{t-7})$, or a state predictive controller is employed. The former method increases the complexity of a policy significantly, while the latter method requires a model $f$.

*Inaccuracies of Command Execution*: Given an acceleration command $u$, the robot will execute something close to $u$, but not $u$ exactly. Thus, learning a function which includes $u$, e.g., the dynamics model (7), can be dangerous since the mapping $(x, \dot{x}, \theta, \dot{\theta}, u) \rightarrow (\ddot{x}, \ddot{\theta})$ is contaminated by the nonlinear dynamics of the robot arm. Indeed, it turned out that we could not learn such a model reliably. This could be remedied by "observing" the command $u$, i.e., by extracting $u = \ddot{x}$ from visual feedback.

*Internal State in Demonstrated Policy*: Investigations with human subjects have shown that humans use internal state in pole balancing. Thus, a policy cannot be observed that easily anymore as claimed in Section 2.2: a regression analysis for extracting the policy of a teacher must find the appropriate time-alignment of observed current state and command(s) in the past. This can become a numerically involved process as regressing a policy based on delayed commands is endangered by singular regression matrices. Consequently, it easily happens that one extracts a *nonstabilizing* policy from the demonstration, which prevents the application of $Q$–learning and $V$–learning as described in Section 2.2.

As a result of these considerations, the most trustworthy item to extract from a demonstration is the model of the pole dynamics. In our implementation it was used in two ways, for calculating the policy as in (6), and in state-predictive control with a Kalman filter to overcome the delays in visual information processing. The model was learned incrementally in real-time by an implementation of RFWR (Schaal & Atkeson 1996). Figure 6 shows the results of learning from scratch and learning from demonstration of the actual robot. Without a demonstration, it took about 10-20 trials before learning succeeded in reliable performance longer than one minute. With a 30 second long demonstration, learning was reliably accomplished in one *single* trial, using a large variety of physically different poles and using demonstrations from arbitrary people in the laboratory.

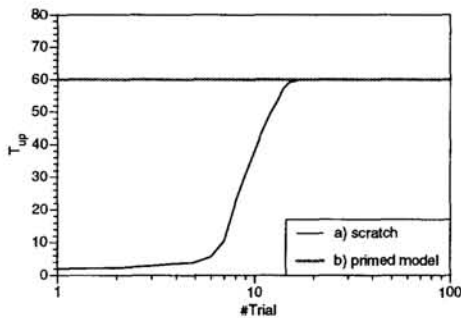

Figure 6: Smoothed average of 10 learning curves of the robot for pole balancing. The trials were aborted after successful balancing of 60 seconds. We also tested long term performance of the learning system by running pole balancing for over an hour—the pole was never dropped.

# 3. CONCLUSION

We discussed learning from demonstration in the context of reinforcement learning, focusing on $Q$–learning, value function learning, and model based reinforcement learning. $Q$–learning and value function learning can theoretically profit from a demonstration by extracting a policy, by using the demonstration data to prime the $Q$/value function, or, in the case of value function learning, by extracting a predictive model of the world. Only in the special case of LQR problems, however, could we find a significant benefit of priming the learner from the demonstration. In contrast, model-based reinforcement learning was able to greatly profit from the demonstration by using the predictive model of the world for "mental simulations". In an implementation with an anthropomorphic robot arm, we illustrated that even in LQR problems, model-based reinforcement learning offers larger robustness towards the complexity in real learning systems than $Q$–learning and value function learning. Using a model-based strategy, our robot learned pole-balancing from a demonstration in a *single* trial with great reliability. The important message of this work is that not every learning approach is equally suited to allow knowledge transfer and/or the incorporation of biases. This issue may serve as a critical additional constraint to evaluate artificial and biological models of learning.

## Acknowledgments

Support was provided by the ATR Human Information Processing Labs, the German Research Association, the Alexander v. Humboldt Foundation, and the German Scholarship Foundation.

## References

Atkeson, C. G. (1994). "Using local trajectory optimizers to speed up global optimization in dynamic programming." In: Moody, Hanson, & Lippmann (Ed.), *Adv. in Neural Inf. Proc. Sys. 6.* Morgan Kaufmann.

Bakker, P., & Kuniyoshi, Y. (1996). "Robot see, robot do: An overview of robot imitation.", Autonomous Systems Section, Electrotechnical Laboratory, Tsukuba Science City, Japan.

Barto, A. G., Sutton, R. S., & Anderson, C. W. (1983). "Neuronlike adaptive elements that can solve difficult learning control problems." *IEEE Transactions on Systems, Man, and Cybernetics,* **SMC-13**, 5.

Bradtke, S. J. (1993). "Reinforcement learning applied to linear quadratic regulation." In: Hanson, J. S., Cowan, J. D., & Giles, C. L. (Eds.), *Advances in Neural Inf. Processing Systems 5,* pp.295-302. Morgan Kaufmann.

Dillmann, R., Kaiser, M., & Ude, A. (1995). "Acquisition of elementary robot skills from human demonstration." In: *International Symposium on Intelligent Robotic Systems (SIRS'95),* Pisa, Italy.

Doya, K. (1996). "Temporal difference learning in continuous time and space." In: Touretzky, D. S., Mozer, M. C., & Hasselmo, M. E. (Eds.), *Advances in Neural Information Processing Systems 8.* MIT Press.

Dufay, B., & Latombe, J.-C. (1984). "An approach to automatic robot programming based on inductive learning." In: Brady, M., & Paul, R. (Eds.), *Robotics Research,* pp.97-115. Cambridge, MA: MIT Press.

Dyer, P., & McReynolds, S. R. (1970). *The computation and theory of opitmal control.* NY: Academic Press.

Ikeuchi, K. (1993b). "Assembly plan from observation.", School of Computer Science, Carnegie Mellon University, Pittsburgh, PA.

Kawato, M., Gandolfo, F., Gomi, H., & Wada, Y. (1994b). "Teaching by showing in kendama based on optimization principle." In: *Proceedings of the International Conference on Artificial Neural Networks (ICANN'94),* **1**, pp.601-606.

Lozano-Perez, T. (1982). "Task-Planning." In: Brady, M., Hollerbach, J. M., Johnson, T. L., Lozano-P_rez, T., & Mason, M. T. (Eds.), , pp.473-498. MIT Press.

Miyamoto, H., Schaal, S., Gandolfo, F., Koike, Y., Osu, R., Nakano, E., Wada, Y., & Kawato, M. (in press). "A Kendama learning robot based on bi-directional theory." *Neural Networks.*

Moore, A. (1991a). "Fast, robust adaptive control by learning only forward models." In: Moody, J. E., Hanson, S. J., & and Lippmann, R. P. (Eds.), *Advances in Neural Inf. Proc. Systems 4.* Morgan Kaufmann.

Schaal, S., & Atkeson, C. G. (1996). "From isolation to cooperation: An alternative of a system of experts." In: Touretzky, D. S., Mozer, M. C., & Hasselmo, M. E. (Eds.), *Advances in Neural Information Processing Systems 8.* Cambridge, MA: MIT Press.

Segre, A. B., & DeJong, G. (1985). "Explanation-based manipulator learning: Acquisition of planning ability through observation." In: *Conference on Robotics and Automation,* pp.555-560.

Singh, S. P., & Sutton, R. S. (1996). "Reinforcement learning with eligibility traces." *Machine Learning.*

Sutton, R. S. (1990). "Integrated architectures for learning, planning, and reacting based on approximating dynamic programming." In: *Proceedings of the International Machine Learning Conference.*

Watkins, C. J. C. H. (1989). "Learning with delayed rewards." Ph.D. thesis, Cambridge University (UK), .

Widrow, B., & Smith, F. W. (1964). "Pattern recognizing control systems." In: *1963 Comp. and Inf. Sciences (COINS) Symp. Proc.,* 288-317, Washington: Spartan.